# A Charge-Based CMOS Parallel Analog Vector Quantizer

**Gert Cauwenberghs**
Johns Hopkins University
ECE Department
3400 N. Charles St.
Baltimore, MD 21218-2686
gert@jhunix.hcf.jhu.edu

**Volnei Pedroni**
California Institute of Technology
EE Department
Mail Code 128-95
Pasadena, CA 91125
pedroni@romeo.caltech.edu

## Abstract

We present an analog VLSI chip for parallel analog vector quantization. The MOSIS 2.0 $\mu$m double-poly CMOS Tiny chip contains an array of $16 \times 16$ charge-based distance estimation cells, implementing a mean absolute difference (MAD) metric operating on a 16-input analog vector field and 16 analog template vectors. The distance cell including dynamic template storage measures $60 \times 78$ $\mu$m$^2$. Additionally, the chip features a winner-take-all (WTA) output circuit of linear complexity, with global positive feedback for fast and decisive settling of a single winner output. Experimental results on the complete $16 \times 16$ VQ system demonstrate correct operation with 34 dB analog input dynamic range and 3 $\mu$sec cycle time at 0.7 mW power dissipation.

## 1 Introduction

Vector quantization (VQ) [1] is a common ingredient in signal processing, for applications of pattern recognition and data compression in vision, speech and beyond. Certain neural network models for pattern recognition, such as Kohonen feature map classifiers [2], are closely related to VQ as well. The implementation of VQ, in its basic form, involves a search among a set of vector templates for the one which best matches the input vector, whereby the degree of matching is quantified by a given vector distance metric. Effi-

cient hardware implementation requires a parallel search over the template set and a fast selection and encoding of the "winning" template. The chip presented here implements a parallel synchronous analog vector quantizer with 16 analog input vector components and 16 dynamically stored analog template vectors, producing a 4-bit digital output word encoding the winning template upon presentation of an input vector. The architecture is fully scalable as in previous implementations of analog vector quantizers, *e.g.* [3,4,5,6], and can be readily expanded toward a larger number of vector components and template vectors without structural modification of the layout. Distinct features of the present implementation include a linear winner-take-all (WTA) structure with globalized positive feedback for fast selection of the winning template, and a mean absolute difference (MAD) metric for the distance estimations, both realized with a minimum amount of circuitry. Using a linear charge-based circuit topology for MAD distance accumulation, a wide voltage range for the analog inputs and templates is achieved at relatively low energy consumption per computation cycle.

## 2   System Architecture

The core of the VQ consists of a $16 \times 16$ 2-D array of distance estimation cells, configured to interconnect columns and rows according to the vector input components and template outputs. Each cell computes in parallel the absolute difference distance between one component $x_j$ of the input vector $\mathbf{x}$ and the corresponding component $y^i{}_j$ of one of the template vectors $\mathbf{y}^i$,

$$d(x_j, y^i{}_j) = |x_j - y^i{}_j| \, , \quad i, j = 1 \dots 16 \, . \tag{1}$$

The mean absolute difference (MAD) distance between input and template vectors is accumulated along rows

$$\hat{d}(\mathbf{x}, \mathbf{y}^i) = \frac{1}{16} \sum_{j=1}^{16} |x_j - y^i{}_j| \, , \quad i = 1 \dots 16 \tag{2}$$

and presented to the WTA, which selects the single winner

$$k^{\text{WTA}} = \arg \min_i \hat{d}(\mathbf{x}, \mathbf{y}^i) \, . \tag{3}$$

Additional parts are included in the architecture for binary encoding of the winning output, and for address selection to write and refresh the template vectors.

## 3   VLSI Circuit Implementation

The circuit implementation of the major components of the VQ, for MAD distance estimation and WTA selection, is described below. Both MAD distance and WTA cells operate in clocked synchronous mode using a precharge/evaluate scheme in the voltage domain. The approach followed here offers a wide analog voltage range of inputs and templates at low power weak-inversion MOS operation, and a fast and decisive settling of the winning output using a single communication line for global positive feedback. The output encoding and address decoding circuitry are implemented using standard CMOS logic.

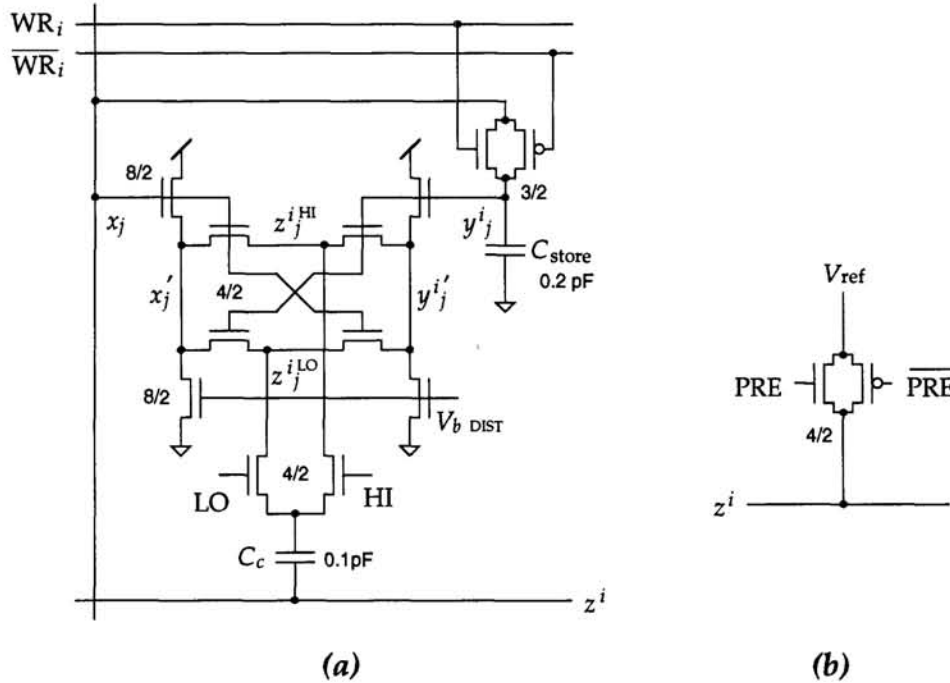

**(a)**                                    **(b)**

Figure 1:   Schematic of distance estimation circuitry.   *(a)* Absolute distance cell.   *(b)* Output precharge circuitry.

### 3.1  Distance Estimation Cell

The schematic of the distance estimation cell, replicated along rows and columns of the VQ array, is shown in Figure 1 (a). The cell contains two source followers, which buffer the input voltage $x_j$ and the template voltage $y^i_j$. The template voltage is stored dynamically onto $C_{\text{store}}$, written or refreshed by activating $\text{WR}_i$ while the $y^i_j$ value is presented on the $x_j$ input line. The $\text{WR}_i$ and $\overline{\text{WR}_i}$ signal levels along rows of the VQ array are driven by the address decoder, which selects a single template vector $\mathbf{y}^i$ to be written to with data presented at the input $\mathbf{x}$ when WR is active.

Additional lateral transistors connect symmetrically to the source follower outputs $x_j{}'$ and $y^i_j{}'$. By means of resistive division, the lateral transistors construct the maximum and minimum of $x_j{}'$ and $y^i_j{}'$ on $z^i_j{}^{\text{HI}}$ and $z^i_j{}^{\text{LO}}$, respectively. In particular, when $x_j$ is much larger than $y^i_j$, the voltage $z^i_j{}^{\text{HI}}$ approaches $x_j{}'$ and the voltage $z^i_j{}^{\text{LO}}$ approaches $y^i_j{}'$. By symmetry, the complementary argument holds in case $x_j$ is much smaller than $y^i_j$. Therefore, the differential component of $z^i_j{}^{\text{HI}}$ and $z^i_j{}^{\text{LO}}$ approximately represents the absolute difference value of $x_j$ and $y^i_j$:

$$
\begin{aligned}
z^i_j{}^{\text{HI}} - z^i_j{}^{\text{LO}} &\approx \max(x_j{}', y^i_j{}') - \min(x_j{}', y^i_j{}') \\
&= |x_j{}' - y^i_j{}'| \approx \kappa \, |x_j - y^i_j| \,,
\end{aligned}
\tag{4}
$$

with $\kappa$ the MOS back gate effect coefficient [7].

The mean absolute difference (MAD) distances (2) are obtained by accumulating con-

tributions (4) along rows of cells through capacitive coupling, using the well known technique of correlated double sampling. To this purpose, a coupling capacitor $C_c$ is provided in every cell, coupling its differential output to the corresponding output row line. In the precharge phase, the maximum values $z^i{}_j{}^{\text{HI}}$ are coupled to the output by activating HI, and the output lines are preset to reference voltage $V_{\text{ref}}$ by activating PRE, Figure 1 (b). In the evaluate phase, PRE is de-activated, and the minimum values $z^i{}_j{}^{\text{LO}}$ are coupled to the output by activating LO. From (4), the resulting voltage outputs on the floating row lines are given by

$$z^i \;=\; V_{\text{ref}} - \frac{1}{16}\sum_{j=1}^{16}(z^i{}_j{}^{\text{HI}} - z^i{}_j{}^{\text{LO}}) \tag{5}$$

$$\approx\; V_{\text{ref}} - \kappa\,\frac{1}{16}\sum_{j=1}^{16}|x_j - y^i{}_j|\;.$$

The last term in (5) corresponds directly to the distance measure $\hat{d}(\mathbf{x}, \mathbf{y}^i)$ in (2). Notice that the negative sign in (5) could be reversed by interchanging clocks HI and LO, if needed. Since the subsequent WTA stage searches for maximum $z^i$, the inverted distance metric is in the form needed for VQ.

Characteristics of the MAD distance estimation (5), measured directly on the VQ array with uniform inputs $x_j$ and templates $y^i{}_j$, are shown in Figure 2. The magnified view in Figure 2 (b) clearly illustrates the effective smoothing of the absolute difference function (4) near the origin, $x_j \approx y^i{}_j$. The smoothing is caused by the shift in $x_j{}'$ and $y^i{}_j{}'$ due to the conductance of the lateral coupling transistors connected to the source follower outputs in Figure 1 (a), and extends over a voltage range comparable to the thermal voltage $kT/q$ depending on the relative geometry of the transistors and current bias level of the source followers. The observed width of the flat region in Figure 2 spans roughly 60 mV, and shows little variation for bias current settings below 0.5 $\mu$A. Tuning of the bias current allows to balance speed and power dissipation requirements, since the output response is slew-rate limited by the source followers.

## 3.2   Winner-Take-All Circuitry

The circuit implementation of the winner-take-all (WTA) function combines the compact sizing and modularity of a linear architecture as in [4,8,9] with positive feedback for fast and decisive output settling independent of signal levels, as in [6,3]. Typical positive feedback structures for WTA operation use a logarithmic tree [6] or a fully interconnected network [3], with implementation complexities of order $O(n\log n)$ and $O(n^2)$ respectively, $n$ being the number of WTA inputs. The present implementation features an $O(n)$ complexity in a linear structure by means of globalized positive feedback, communicated over a single line.

The schematic of the WTA cell, receiving the input $z^i$ and constructing the digital output $d_i$ through global competition communicated over the COMM line, is shown in Figure 3. The global COMM line is source connected to input transistor Mi and positive feedback transistor Mf, and receives a constant bias current $I_{\text{b (WTA)}}$ from Mb1. Locally, the WTA operation is governed by the dynamics of $d_i{}'$ on (parasitic) capacitor $C_p$. A high pulse

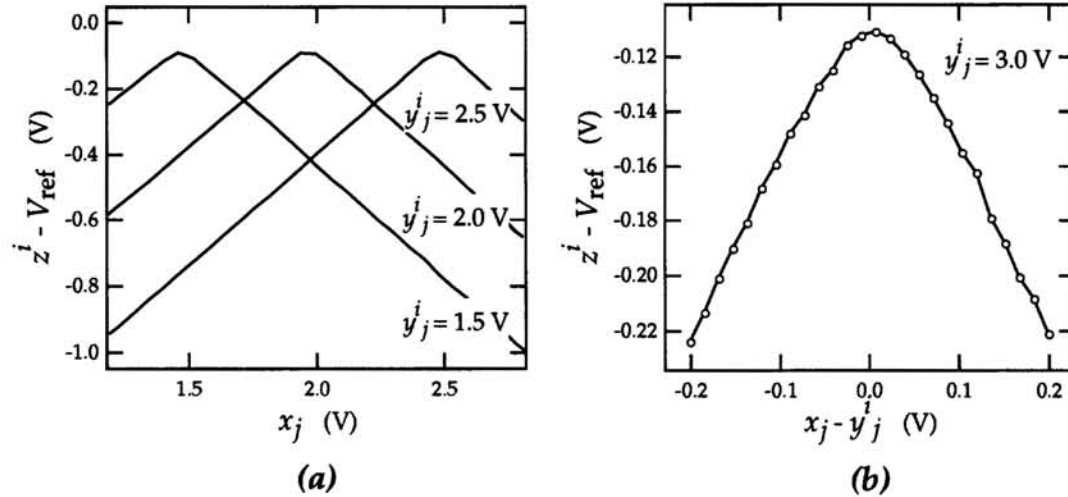

Figure 2:  Distance estimation characteristics,  *(a)* for various values of $y^i{}_j$; *(b)* magnified view.

on RST, resetting $d_i'$ to zero, marks the beginning of the WTA cycle. With Mf initially inactive, the total bias current $n\,I_{b\,\text{(WTA)}}$ through COMM is divided over all competing WTA cells, according to the relative $z^i$ voltage levels, and each cell fraction is locally mirrored by the Mm1-Mm2 pair onto $d_i'$, charging $C_p$. The cell with the highest $z^i$ input voltage receives the largest fraction of bias current, and charges $C_p$ at the highest rate. The winning output is determined by the first $d_i'$ reaching the threshold to turn on the corresponding Mf feedback transistor, say $i = k$. This threshold voltage is given by the source voltage on COMM, common for all cells. The positive feedback of the state $d_k'$ through Mf, which eventually claims the entire fraction of the bias current, enhances and latches the winning output level $d_k'$ to the positive supply and shuts off the remaining losing outputs $d_i'$ to zero, $i \neq k$. The additional circuitry at the output stage of the cell serves to buffer the binary $d_i'$ value at the $d_i$ output terminal.

No more than one winner can practically co-exist at equilibrium, by nature of the combined positive feedback and global renormalization in the WTA competition. Moreover, the output settling times of the winner and losers are fairly independent of the input signal levels, and are given mainly by the bias current level $I_{b\,\text{(WTA)}}$ and the parasitic capacitance $C_p$. Tests conducted on a separate 16-element WTA array, identical to the one used on the VQ chip, have demonstrated single-winner WTA operation with response time below 0.5 $\mu$sec at less than 2 $\mu$W power dissipation per cell.

## 4  Functionality Test

To characterize the performance of the entire VQ system under typical real-time conditions, the chip was presented a periodic sequence of 16 distinct input vectors $\mathbf{x}(i)$, stored and refreshed dynamically in the 16 template locations $\mathbf{y}^i$ by circularly incrementing the template address and activating WR at the beginning of every cycle. The test vectors rep-

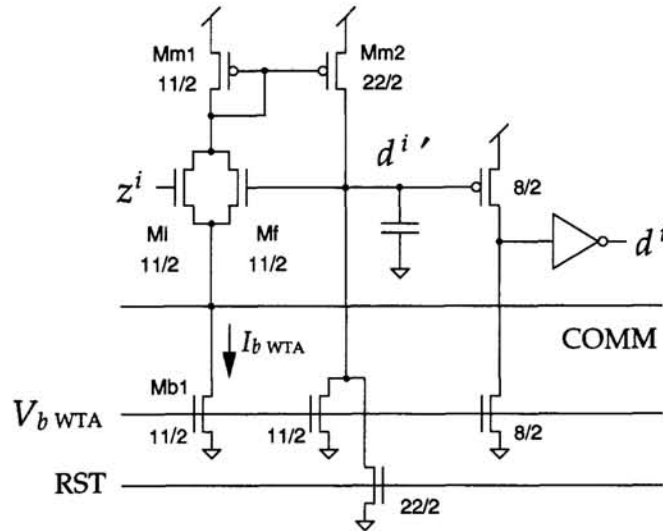

Figure 3:   Circuit schematic of winner-take-all cell.

resent a single triangular pattern rotated over the 16 component indices with single index increments in sequence. The fundamental component $x_0(i)$ is illustrated on the top trace of the scope plot in Figure 4. The other components are uniformly displaced in time over one period, by a number of cycles equal to the index, $x_j(i) = x_0(i - j \bmod 16)$. Figure 4 also displays the VQ output waveforms in response to the triangular input sequence, with the desired parabolic profile for the analog distance output $z^0$ and the expected alternating bit pattern of the WTA least significant output bit.[1] The triangle test performed correctly at speeds limited by the instrumentation equipment, and the dissipated power on the chip measures 0.7 mW at 3 $\mu$sec cycle time[2] and 5 V supply voltage.

An estimate for the dynamic range of analog input and template voltages was obtained directly by observing the smallest and largest absolute voltage difference still resolved correctly by the VQ output, uniformly over all components. By tuning the voltage range of the triangular test vectors, the recorded minimum and maximum voltage amplitudes for 5 V supply voltage are $V_{\min} = 87.5$ mV and $V_{\max} = 4$ V, respectively. The estimated analog dynamic range $V_{\max}/V_{\min}$ is thus 45.7, or roughly 34 dB, per cell. The value obtained for $V_{\min}$ indicates that the dynamic range is limited mainly by the smoothing of the absolute distance measure characteristic (1) near the origin in Figure 2 (b). We notice that a similar limitation of dynamic range applies to other distance metrics with vanishing slope near the origin as well, the popular mean square error (MSE) formulation in particular. The MSE metric is frequently adopted in VQ implementations using strong inversion MOS circuitry, and offers a dynamic range typically worse than obtained here regardless of implementation accuracy, due to the relatively wide flat region of the MSE distance function near the origin.

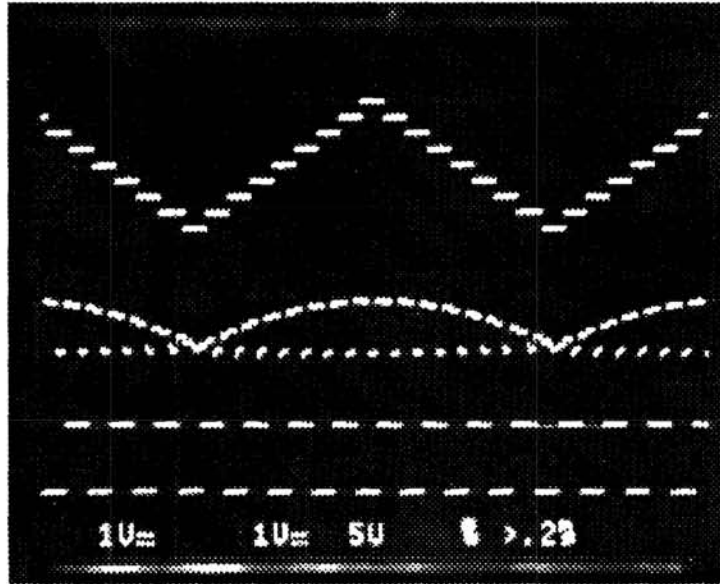

Figure 4: Scope plot of VQ waveforms. Top: Analog input $x_0$. Center: Analog distance output $z^0$. Bottom: Least significant bit of encoded output.

Table 1: Features of the VQ chip

| | |
|---|---|
| Technology | 2 μm p-well double-poly CMOS |
| Supply voltage | + 5 V |
| Power dissipation | |
|     VQ chip | 0.7 mW (3 μsec cycle time) |
| Dynamic range | |
|     inputs, templates | 34 dB |
| Area | |
|     VQ chip | 2.2 mm X 2.25 mm |
|     distance cell | 60 μm X 78 μm |
|     WTA cell | 76 μm X 80 μm |

## 5 Conclusion

We proposed and demonstrated a synchronous charge-based CMOS VLSI system for parallel analog vector quantization, featuring a mean absolute difference (MAD) metric, and a linear winner-take-all (WTA) structure with globalized positive feedback. By virtue of the MAD metric, a fairly large (34 dB) analog dynamic range of inputs and templates has been obtained in the distance computations through simple charge-based circuitry. Likewise, fast and unambiguous settling of the WTA outputs, using global competition communicated over a single wire, has been obtained by adopting a compact linear circuit structure to implement the positive feedback WTA function. The resulting structure of the VQ chip is highly modular, and the functional characteristics are fairly consistent over a wide range of bias levels, including the MOS weak inversion and subthreshold regions. This allows the circuitry to be tuned to accommodate various speed and power requirements. A summary of the chip features of the $16 \times 16$ vector quantizer is presented in Table I.

### Acknowledgments

Fabrication of the CMOS chip was provided through the DARPA/NSF MOSIS service. The authors thank Amnon Yariv for stimulating discussions and encouragement.

## Footnotes

[1]The voltages on the scope plot are inverted as a consequence of the chip test setup.

[2]including template write operations

### References

[1] A. Gersho and R.M. Gray, *Vector Quantization and Signal Compression,* Norwell, MA: Kluwer, 1992.

[2] T. Kohonen, *Self-Organisation and Associative Memory,* Berlin: Springer-Verlag, 1984.

[3] Y. He and U. Cilingiroglu, "A Charge-Based On-Chip Adaptation Kohonen Neural Network," *IEEE Transactions on Neural Networks,* vol. **4** (3), pp 462-469, 1993.

[4] J.C. Lee, B.J. Sheu, and W.C. Fang, "VLSI Neuroprocessors for Video Motion Detection," *IEEE Transactions on Neural Networks,* vol. **4** (2), pp 78-191, 1993.

[5] R. Tawel, "Real-Time Focal-Plane Image Compression," in *Proceedings Data Compression Conference,*, Snowbird, Utah, IEEE Computer Society Press, pp 401-409, 1993.

[6] G.T. Tuttle, S. Fallahi, and A.A. Abidi, "An 8b CMOS Vector A/D Converter," in *ISSCC Technical Digest,* IEEE Press, vol. **36**, pp 38-39, 1993.

[7] C.A. Mead, *Analog VLSI and Neural Systems,* Reading, MA: Addison-Wesley, 1989.

[8] J. Lazzaro, S. Ryckebusch, M.A. Mahowald, and C.A. Mead, "Winner-Take-All Networks of O(n) Complexity," in *Advances in Neural Information Processing Systems,* San Mateo, CA: Morgan Kaufman, vol. **1**, pp 703-711, 1989.

[9] A.G. Andreou, K.A. Boahen, P.O. Pouliquen, A. Pavasovic, R.E. Jenkins, and K. Strohbehn, "Current-Mode Subthreshold MOS Circuits for Analog VLSI Neural Systems," *IEEE Transactions on Neural Networks,* vol. **2** (2), pp 205-213, 1991.